# Estimating image bases for visual image reconstruction from human brain activity

**Yusuke Fujiwara**[1]    **Yoichi Miyawaki**[2,1]    **Yukiyasu Kamitani**[1]
[1]ATR Computational Neuroscience Laboratories
[2]National Institute of Information and Communications Technology
2-2-2 Hikaridai, Seika-cho, Kyoto, Japan
yureisoul@gmail.com  yoichi_m@atr.jp  kmtn@atr.jp

## Abstract

Image representation based on image bases provides a framework for understanding neural representation of visual perception. A recent fMRI study has shown that arbitrary contrast-defined visual images can be reconstructed from fMRI activity patterns using a combination of multi-scale local image bases. In the reconstruction model, the mapping from an fMRI activity pattern to the contrasts of the image bases was learned from measured fMRI responses to visual images. But the shapes of the images bases were fixed, and thus may not be optimal for reconstruction. Here, we propose a method to build a reconstruction model in which image bases are automatically extracted from the measured data. We constructed a probabilistic model that relates the fMRI activity space to the visual image space via a set of latent variables. The mapping from the latent variables to the visual image space can be regarded as a set of image bases. We found that spatially localized, multi-scale image bases were estimated near the fovea, and that the model using the estimated image bases was able to accurately reconstruct novel visual images. The proposed method provides a means to discover a novel functional mapping between stimuli and brain activity patterns.

## 1   Introduction

The image basis is a key concept for understanding neural representation of visual images. Using image bases, we can consider natural scenes as a combination of simple elements corresponding to neural units. Previous works have shown that image bases similar to receptive fields of simple cells are learned from natural scenes by the sparse coding algorithm [4, 9]. A recent fMRI study has shown that visual images can be reconstructed using a linear combination of multi-scale image bases (1x1, 1x2, 2x1, and 2x2 pixels covering an entire image), whose contrasts were predicted from the fMRI activity pattern [6]. The multi-scale bases produced more accurate reconstruction than the pixel-by-pixel prediction, and each scale contributed to reconstruction in a way consistent with known visual cortical representation. However, the predefined shapes of image bases may not be optimal for image reconstruction.

Here, we developed a method to automatically extract image bases from measured fMRI responses to visual stimuli, and used them for image reconstruction. We employed the framework of canonical correlation analysis (CCA), in which two multi-dimensional observations are related via a common coordinate system. CCA finds multiple correspondences between a weighted sum of voxels and a weighted sum of pixels. These correspondences provide an efficient mapping between the two observations. The pixel weights for each correspondence can be thought to define an image basis.

As the early visual cortex is known to be organized in a retinotopic manner, one can assume that a small set of pixels corresponds to a small set of voxels. To facilitate the mapping between small

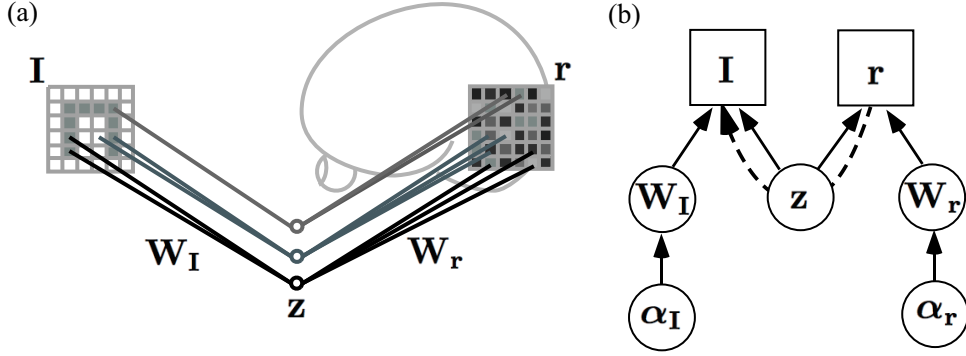

(a) (b)

Figure 1: **Model for estimating image bases.** (a) Illustration of the model framework. The visual image $\mathbf{I}$ (pixels) and an fMRI activity pattern $\mathbf{r}$ (voxels) is linked by latent variables $\mathbf{z}$. The links from each latent variable to image pixels define an image basis $\mathbf{W_I}$, and the links from each latent variable to fMRI voxels is called a weight vector $\mathbf{W_r}$. (b) Graphical representation of the model. Circles indicate model parameters to be estimated and squares indicate observations. The matrices $\mathbf{W_I}$ and $\mathbf{W_r}$, the common latent variable $\mathbf{z}$, and the inverse variances $\boldsymbol{\alpha_I}$ and $\boldsymbol{\alpha_r}$ are simultaneously estimated using the variational Bayesian method. Using the estimated parameters, the predictive distribution for a visual image given a new brain activity pattern is constructed (dashed line).

sets of pixels and voxels, we extended CCA to Bayesian CCA [10] with sparseness priors. Bayesian CCA treats the multiple correspondences as latent variables with two transformation matrices to two sets of observations. The transformation matrix to the visual image can be regarded as a set of image bases. The matrices are assumed to be random variables with hyper-parameters. We introduced a sparseness prior into each element of the matrices, such that only small subsets of voxels and pixels are related with non-zero matrix elements.

The Bayesian CCA model was applied to the data set of Miyawaki et al. [6]. We show that spatially localized image bases were extracted, especially around the foveal region, whose shapes were similar to those used in the previous work. We also demonstrate that the model using the estimated image bases produced accurate visual image reconstruction.

## 2   Method

We constructed a model in which a visual image is related with an fMRI activity pattern via latent variables (Figure 1). Each latent variable has links to a set of pixels, which can be regarded as an image basis because links from a single latent variable construct an element of a visual image. The latent variable also has multiple links to a set of fMRI voxels, which we call a weight vector. This model is equivalent to CCA: each latent variable corresponds to a canonical coefficient [3] that bundles a subset of fMRI voxels responding to a specific visual stimulus. We then extended the CCA model to the Bayesian CCA model that can conduct a sparse selection of these links automatically.

### 2.1   Canonical Correlation Analysis

We first consider the standard CCA for estimating image bases given visual images $\mathbf{I}$ and fMRI activity patterns $\mathbf{r}$. Let $\mathbf{I}$ be an $N \times 1$ vector and $\mathbf{r}$ be a $K \times 1$ vector where $N$ is the number of image pixels, $K$ is the number of fMRI voxels and $t$ is a sample index. Both data sets are independent identically distributed (i.i.d.) samples. CCA finds linear combinations $u_1(t) = \mathbf{a}_1' \cdot \mathbf{I}(t)$ and $v_1(t) = \mathbf{b}_1' \cdot \mathbf{r}(t)$ such that the correlation between $u_1$ and $v_1$ is maximized. The variables $u_1$ and $v_1$ are called the first canonical variables and the vectors $\mathbf{a}_1$ and $\mathbf{b}_1$ are called the canonical coefficients. Then, the second canonical variables $u_2(t) = \mathbf{a}_2' \cdot \mathbf{I}(t)$ and $v_2(t) = \mathbf{b}_2' \cdot \mathbf{r}(t)$ are sought by maximizing the correlation of $u_2$ and $v_2$ while the second canonical variables are orthogonalized to the first canonical variables. This procedure is continued up to a pre-defined number of times $M$. The number $M$ is conventionally set to the smaller dimension of the two sets of observations: in our case, $M = N$ because the number of visual-image pixels is much smaller than that of the fMRI

voxels ($N < K$). The $M$ sets of canonical variables are summarized as

$$\mathbf{u}(t) = \mathbf{A} \cdot \mathbf{I}(t), \tag{1}$$

$$\mathbf{v}(t) = \mathbf{B} \cdot \mathbf{r}(t), \tag{2}$$

where $\mathbf{u}(t)$ and $\mathbf{v}(t)$ are $M \times 1$ vectors, $\mathbf{A}$ is an $M \times N$ matrix, and $\mathbf{B}$ is a $M \times K$ matrix. The matrices $\mathbf{A}$ and $\mathbf{B}$ are obtained by solving the eigen problem of the covariance matrix between $\mathbf{I}$ and $\mathbf{r}$ [1]. The visual image can be reconstructed by

$$\mathbf{I}(t) = \mathbf{A}^{-1} \cdot \mathbf{B} \cdot \mathbf{r}(t), \tag{3}$$

where each column vector of the inverse matrix $\mathbf{A}^{-1}$ is an image basis.

## 2.2 Bayesian CCA

Bayesian CCA introduces common latent variables that relate a visual image $\mathbf{I}$ and the fMRI activity pattern $\mathbf{r}$ with image basis set $\mathbf{W_I}$ and weight vector set $\mathbf{W_r}$ (Figure 1 (b)). These variables are treated as random variables and prior distributions are assumed for each variable. Hyper-prior distributions are also assumed for an inverse variance of each element of the image bases and the weight vectors. The image bases and the weight vectors are estimated as a posterior distribution by the variational Bayesian method [2]. After the parameters are determined, a predictive distribution for the visual image can be calculated.

We assume two likelihood functions. One is for visual images that are generated from latent variables. The other is for fMRI activity patterns that are generated from the same latent variables. When observation noises for visual images and fMRI voxels are assumed to follow a Gaussian distribution with zero mean and spherical covariance, the likelihood functions of the visual image $\mathbf{I}$ and the fMRI activity pattern $\mathbf{r}$ are

$$P(\mathbf{I}|\mathbf{W_I}, \mathbf{z}) \propto \exp\left[ -\frac{1}{2}\beta_\mathbf{I} \sum_{t=1}^{T} ||\mathbf{I}(t) - \mathbf{W_I} \cdot \mathbf{z}(t)||^2 \right], \tag{4}$$

$$P(\mathbf{r}|\mathbf{W_r}, \mathbf{z}) \propto \exp\left[ -\frac{1}{2}\beta_\mathbf{r} \sum_{t=1}^{T} ||\mathbf{r}(t) - \mathbf{W_r} \cdot \mathbf{z}(t)||^2 \right], \tag{5}$$

where $\mathbf{W_I}$ is an $N \times M$ matrix representing $M$ image bases, each of which consists of $N$ pixels, $\mathbf{W_r}$ is a $K \times M$ matrix representing $M$ weight vectors, each of which consist of $K$ voxels, $\mathbf{z}(t)$ is an $M \times 1$ vector representing latent variables, $\beta_\mathbf{I}^{-1}$ and $\beta_\mathbf{r}^{-1}$ are scalar variables representing unknown noise variances of the visual image and fMRI activity pattern, and $T$ is the number of observations.

The latent variables are treated as the following Gaussian prior distribution,

$$P_0(\mathbf{z}) \propto \exp\left[ -\frac{1}{2} \sum_{t=1}^{T} ||\mathbf{z}(t)||^2 \right]. \tag{6}$$

The image bases and weight vectors are regarded as random variables, and the prior distributions of them are assumed as,

$$P_0(\mathbf{W_I}|\boldsymbol{\alpha_I}) \propto \exp\left[ -\frac{1}{2} \sum_{n=1}^{N} \sum_{m=1}^{M} \boldsymbol{\alpha}_{\mathbf{I}(n,m)} \big( \mathbf{W}_{\mathbf{I}(n,m)} \big)^2 \right], \tag{7}$$

$$P_0(\mathbf{W_r}|\boldsymbol{\alpha_r}) \propto \exp\left[ -\frac{1}{2} \sum_{k=1}^{K} \sum_{m=1}^{M} \boldsymbol{\alpha}_{\mathbf{r}(k,m)} \big( \mathbf{W}_{\mathbf{r}(k,m)} \big)^2 \right], \tag{8}$$

where $\boldsymbol{\alpha}_{\mathbf{I}(n,m)}$ and $\boldsymbol{\alpha}_{\mathbf{r}(k,m)}$ are the inverse variances of the elements in $\mathbf{W_I}$ and $\mathbf{W_r}$, respectively, which are assumed to be mutually independent.

We also assume hyper-prior distributions for the inverse variances $\boldsymbol{\alpha}_{\mathbf{I}(n,m)}$ and $\boldsymbol{\alpha}_{\mathbf{r}(k,m)}$,

$$P_0(\boldsymbol{\alpha_I}) = \prod_n \prod_m \mathcal{G}(\boldsymbol{\alpha}_{\mathbf{I}(n,m)}|\bar{\boldsymbol{\alpha}}_{\mathbf{I}(n,m)}, \boldsymbol{\gamma}_{\mathbf{I}(n,m)}), \tag{9}$$

$$P_0(\boldsymbol{\alpha_r}) = \prod_k \prod_m \mathcal{G}(\boldsymbol{\alpha}_{\mathbf{I}(k,m)}|\bar{\boldsymbol{\alpha}}_{\mathbf{r}(k,m)}, \boldsymbol{\gamma}_{\mathbf{r}(k,m)}), \tag{10}$$

where $\mathcal{G}(\alpha|\bar{\alpha}, \gamma)$ represents the Gamma distribution with mean $\bar{\alpha}$ and confidence parameter $\gamma$. For our analysis, all the means $\bar{\boldsymbol{\alpha}}_{\mathbf{I}(n,m)}$ and $\bar{\boldsymbol{\alpha}}_{\mathbf{r}(k,m)}$ were set to 1 and all the confidence parameters $\boldsymbol{\gamma}_{\mathbf{I}(n,m)}$ and $\boldsymbol{\gamma}_{\mathbf{r}(k,m)}$ were set to 0.

This configuration of the prior and hyper-prior settings is known as the automatic relevance determination (ARD), where non-effective parameters are automatically driven to zero [7]. In the current case, these priors and hyper-priors lead to a sparse selection of links from each latent variable to pixels and voxels.

Prior distributions of observation noise are assumed as non-informative priors, which are described by the observation noise,

$$P_0(\beta_{\mathbf{I}}) = \frac{1}{\beta_{\mathbf{I}}}, \tag{11}$$

$$P_0(\beta_{\mathbf{r}}) = \frac{1}{\beta_{\mathbf{r}}}. \tag{12}$$

## 2.3 Parameter estimation by the variational Bayesian method

The image bases and weight vectors are estimated as a posterior distribution $P(\mathbf{W_I}, \mathbf{W_r}|\mathbf{I}, \mathbf{r})$, given the likelihood functions (Eqs. (4) and (5)), the prior distributions (Eqs. (6) - (8), (11) and (12)), and the hyper-prior distributions (Eqs. (9) and (10)). This posterior distribution is obtained by marginalizing the joint posterior distribution $P(\mathbf{W_I}, \mathbf{W_r}, \mathbf{z}, \boldsymbol{\alpha_I}, \boldsymbol{\alpha_r}, \beta_{\mathbf{I}}, \beta_{\mathbf{r}}|\mathbf{I}, \mathbf{r})$ with respect to latent variables and variance parameters,

$$P(\mathbf{W_I}, \mathbf{W_r}|\mathbf{I}, \mathbf{r}) = \int \mathrm{d}\mathbf{z}\mathrm{d}\boldsymbol{\alpha_I}\mathrm{d}\boldsymbol{\alpha_r}\mathrm{d}\beta_{\mathbf{I}}\mathrm{d}\beta_{\mathbf{r}} P(\mathbf{W_I}, \mathbf{W_r}, \mathbf{z}, \boldsymbol{\alpha_I}, \boldsymbol{\alpha_r}, \beta_{\mathbf{I}}, \beta_{\mathbf{r}}|\mathbf{I}, \mathbf{r}). \tag{13}$$

Since the joint posterior distribution cannot be calculated analytically, we approximate it using a trial distribution based on the variational Bayesian (VB) method [2]. In the VB method, a trial distribution $\mathcal{Q}(\mathbf{W_I}, \mathbf{W_r}, \mathbf{z}, \boldsymbol{\alpha_I}, \boldsymbol{\alpha_r}, \beta_{\mathbf{I}}, \beta_{\mathbf{r}})$ with the following factorization is assumed,

$$\mathcal{Q}(\mathbf{W_I}, \mathbf{W_r}, \mathbf{z}, \boldsymbol{\alpha_I}, \boldsymbol{\alpha_r}, \beta_{\mathbf{I}}, \beta_{\mathbf{r}}) = \mathcal{Q}_{\mathbf{w}}(\mathbf{W_I})\mathcal{Q}_{\mathbf{w}}(\mathbf{W_r})\mathcal{Q}_{\mathbf{z}}(\mathbf{z})\mathcal{Q}_{\alpha}(\boldsymbol{\alpha_I}, \boldsymbol{\alpha_r}, \beta_{\mathbf{I}}, \beta_{\mathbf{r}}). \tag{14}$$

The joint posterior distribution $P(\mathbf{W_I}, \mathbf{W_r}, \mathbf{z}, \boldsymbol{\alpha_I}, \boldsymbol{\alpha_r}, \beta_{\mathbf{I}}, \beta_{\mathbf{r}}|\mathbf{I}, \mathbf{r})$ is approximated by the factorized distribution (Eq. (14)). According to the standard calculation of the VB method, the trial distribution of the image bases $\mathcal{Q}_{\mathbf{w}}(\mathbf{W_I})$ is derived as

$$\mathcal{Q}_{\mathbf{w}}(\mathbf{W_I}) = \prod_{n=1}^{N} \prod_{m=1}^{M} \mathcal{N}(\mathbf{W}_{\mathbf{I}(n,m)}|\overline{\mathbf{W}}_{\mathbf{I}(n,m)}, \boldsymbol{\sigma}_{\mathbf{I}(n,m)}^{-1}), \tag{15}$$

where

$$\overline{\mathbf{W}}_{\mathbf{I}(n,m)} = \bar{\beta}_{\mathbf{I}}\boldsymbol{\sigma}_{\mathbf{I}(n,m)}^{-1} \sum_{t=1}^{T} \mathbf{I}_n(t)\mathbf{z}_m(t), \tag{16}$$

$$\boldsymbol{\sigma}_{\mathbf{I}(n,m)} = \bar{\beta}_{\mathbf{I}}\left(\sum_{t=1}^{T} \mathbf{z}_m^2(t) + T\boldsymbol{\Sigma}_{\mathbf{z}(m,m)}^{-1}\right) + \overline{\boldsymbol{\alpha}}_{\mathbf{I}(n,m)}, \tag{17}$$

and $\mathcal{N}(x|\bar{x}, \sigma^{-1})$ represents a Gaussian distribution with mean $\bar{x}$ and variance $\sigma^{-1}$. The trial distribution of the weight vectors $\mathcal{Q}_{\mathbf{w}}(\mathbf{W_r})$ is obtained in a similar way, by replacing $\mathbf{I}$ with $\mathbf{r}$, $n$ with $k$, and $N$ with $K$ in Eqs. (15-17). The trial distribution of the latent variables $\mathcal{Q}_{\mathbf{z}}(\mathbf{z})$ is obtained by

$$\mathcal{Q}_{\mathbf{z}}(\mathbf{z}) = \prod_{t=1}^{T} \mathcal{N}(\mathbf{z}(t)|\bar{\mathbf{z}}(t), \boldsymbol{\Sigma}_{\mathbf{z}}^{-1}), \tag{18}$$

where

$$\bar{\mathbf{z}}(t) = \boldsymbol{\Sigma}_{\mathbf{z}}^{-1}\left(\bar{\beta}_{\mathbf{I}}\overline{\mathbf{W}}_{\mathbf{I}}'\mathbf{I}(t) + \bar{\beta}_{\mathbf{r}}\overline{\mathbf{W}}_{\mathbf{r}}'\mathbf{r}(t)\right), \tag{19}$$

$$\boldsymbol{\Sigma}_{\mathbf{z}} = \bar{\beta}_{\mathbf{I}}\left(\overline{\mathbf{W}}_{\mathbf{I}}'\overline{\mathbf{W}}_{\mathbf{I}} + \boldsymbol{\Sigma}_{\mathbf{w_I}}^{-1}\right) + \bar{\beta}_{\mathbf{r}}\left(\overline{\mathbf{W}}_{\mathbf{r}}'\overline{\mathbf{W}}_{\mathbf{r}} + \boldsymbol{\Sigma}_{\mathbf{w_r}}^{-1}\right) + \mathbf{E}. \tag{20}$$

In Eq. (20), $\mathbf{E}$ is an identity matrix, and $\mathbf{\Sigma}_{\mathbf{w_I}}$ and $\mathbf{\Sigma}_{\mathbf{w_r}}$ are defined as

$$\mathbf{\Sigma}_{\mathbf{w_I}} = \mathrm{diag}\left(\left[\sum_{n=1}^{N}\boldsymbol{\sigma}_{\mathbf{I}(n,1)}, \cdots, \sum_{n=1}^{N}\boldsymbol{\sigma}_{\mathbf{I}(n,M)}\right]\right), \tag{21}$$

$$\mathbf{\Sigma}_{\mathbf{w_r}} = \mathrm{diag}\left(\left[\sum_{k=1}^{K}\boldsymbol{\sigma}_{\mathbf{r}(k,1)}, \cdots, \sum_{k=1}^{K}\boldsymbol{\sigma}_{\mathbf{r}(k,M)}\right]\right). \tag{22}$$

Finally, the distribution of the inverse variances $\mathcal{Q}_\alpha(\boldsymbol{\alpha}_\mathbf{I}, \boldsymbol{\alpha}_\mathbf{r}, \beta_\mathbf{I}, \beta_\mathbf{r})$ is further factorized into $\mathcal{Q}_\alpha(\boldsymbol{\alpha}_\mathbf{I})\mathcal{Q}_\alpha(\boldsymbol{\alpha}_\mathbf{r})\mathcal{Q}_\alpha(\beta_\mathbf{I})\mathcal{Q}_\alpha(\beta_\mathbf{r})$, each having a function form equivalent to a gamma distribution. The expectation of $\boldsymbol{\alpha}_{\mathbf{I}(n,m)}$ is given by

$$\bar{\boldsymbol{\alpha}}_{\mathbf{I}(n,m)} = \left(\frac{1}{2} + \boldsymbol{\gamma}_{\mathbf{I0}(n,m)}\right)\left(\frac{1}{2}(\overline{\mathbf{W}}_{\mathbf{I}(n,m)})^2 + \frac{1}{2}\boldsymbol{\sigma}_{\mathbf{I}(n,m)}^{-1} + \boldsymbol{\gamma}_{\mathbf{I0}(n,m)}\boldsymbol{\alpha}_{\mathbf{I0}(n,m)}^{-1}\right)^{-1}, \tag{23}$$

and that of $\beta_\mathbf{I}$ is given by

$$\bar{\beta}_\mathbf{I} = NT\left\{\sum_{t=1}^{T}||\mathbf{I}(t) - \overline{\mathbf{W}}_\mathbf{I}\bar{\mathbf{z}}(t)||^2 + \mathrm{Tr}\left[\mathbf{\Sigma}_{\mathbf{w_I}}^{-1}\left(\sum_{t=1}^{T}\mathbf{z}(t)\mathbf{z}'(t) + T\mathbf{\Sigma}_\mathbf{z}^{-1}\right) + T\mathbf{\Sigma}_\mathbf{z}^{-1}\overline{\mathbf{W}}_\mathbf{I}'\overline{\mathbf{W}}_\mathbf{I}\right]\right\}^{-1}. \tag{24}$$

The expectations of $\mathcal{Q}_\alpha(\boldsymbol{\alpha}_\mathbf{r})$ and $\mathcal{Q}_\alpha(\beta_\mathbf{r})$ are obtained in a similar way, by replacing $\mathbf{I}$ with $\mathbf{r}$, $n$ with $k$, and $N$ with $K$ in Eq. (23) and Eq. (24), respectively. The expectations of these distributions are used in the calculation of $\mathcal{Q}_\mathbf{w}(\mathbf{W}_\mathbf{I})$, $\mathcal{Q}_\mathbf{w}(\mathbf{W}_\mathbf{r})$ and $\mathcal{Q}_\mathbf{z}(\mathbf{z})$ (Eqs. (15) - (20)). The algorithm estimates the joint posterior by successive calculations of 1) $\mathcal{Q}_\mathbf{w}(\mathbf{W}_\mathbf{I})$ and $\mathcal{Q}_\mathbf{w}(\mathbf{W}_\mathbf{r})$, 2) $\mathcal{Q}_\mathbf{z}(\mathbf{z})$, and 3) $\mathcal{Q}_\alpha(\boldsymbol{\alpha}_\mathbf{I}, \boldsymbol{\alpha}_\mathbf{r}, \beta_\mathbf{I}, \beta_\mathbf{r})$. After the algorithm converges, image bases $\overline{\mathbf{W}}_\mathbf{I}$ are calculated by taking the expectation of $\mathcal{Q}(\mathbf{W}_\mathbf{I})$.

## 2.4 Predictive distribution for visual image reconstruction

Using the estimated parameters, we can derive the predictive distribution for a visual image $\mathbf{I}_{\mathrm{new}}$ given a new brain activity $\mathbf{r}_{\mathrm{new}}$ (Figure 1 (b), dashed line). Note that $\mathbf{I}_{\mathrm{new}}$ and $\mathbf{r}_{\mathrm{new}}$ were taken from the data set reserved for testing the model, independent of the data set to estimate the model parameters. The predictive distribution $P(\mathbf{I}_{\mathrm{new}}|\mathbf{r}_{\mathrm{new}})$ is constructed from the likelihood of the visual image (Eq. (4)), the estimated distribution of image bases $\mathcal{Q}(\mathbf{W}_\mathbf{I})$ (Eqs. (15) - (17)), and a posterior distribution of latent variables $P(\mathbf{z}_{\mathrm{new}}|\mathbf{r}_{\mathrm{new}})$ as follows,

$$P(\mathbf{I}_{\mathrm{new}}|\mathbf{r}_{\mathrm{new}}) = \int d\mathbf{W}_\mathbf{I}d\mathbf{z}_{\mathrm{new}}P(\mathbf{I}_{\mathrm{new}}|\mathbf{W}_\mathbf{I}, \mathbf{z}_{\mathrm{new}})\mathcal{Q}(\mathbf{W}_\mathbf{I})P(\mathbf{z}_{\mathrm{new}}|\mathbf{r}_{\mathrm{new}}). \tag{25}$$

Because the multiple integral over the random variable $\mathbf{W}_\mathbf{I}$ and $\mathbf{z}_{\mathrm{new}}$ is intractable, we replace the random variable $\mathbf{W}_\mathbf{I}$ with the estimated image bases $\overline{\mathbf{W}}_\mathbf{I}$ to vanish the integral over $\mathbf{W}_\mathbf{I}$. Then the predictive distribution becomes

$$P(\mathbf{I}_{\mathrm{new}}|\mathbf{r}_{\mathrm{new}}) \simeq \int d\mathbf{z}_{\mathrm{new}}P(\mathbf{I}_{\mathrm{new}}|\mathbf{z}_{\mathrm{new}})P(\mathbf{z}_{\mathrm{new}}|\mathbf{r}_{\mathrm{new}}), \tag{26}$$

where

$$P(\mathbf{I}_{\mathrm{new}}|\mathbf{z}_{\mathrm{new}}) \propto \exp\left[-\frac{1}{2}\bar{\beta}_\mathbf{I}||\mathbf{I}_{\mathrm{new}} - \overline{\mathbf{W}}_\mathbf{I}\mathbf{z}_{\mathrm{new}}||^2\right]. \tag{27}$$

Since $P(\mathbf{z}_{\mathrm{new}}|\mathbf{r}_{\mathrm{new}})$ is an unknown distribution, we approximate $P(\mathbf{z}_{\mathrm{new}}|\mathbf{r}_{\mathrm{new}})$ based on the trial distribution $\mathcal{Q}(\mathbf{z})$ (Eqs. (18) - (20)). We construct an approximate distribution $\widetilde{\mathcal{Q}}_\mathbf{z}(\mathbf{z}_{\mathrm{new}})$, by omitting the terms related to the visual image in Eqs. (18) - (20),

$$\widetilde{\mathcal{Q}}_\mathbf{z}(\mathbf{z}_{\mathrm{new}}) = \mathcal{N}(\mathbf{z}|\bar{\mathbf{z}}_{\mathrm{new}}, \mathbf{\Sigma}_{\mathbf{z}\mathrm{new}}^{-1}), \tag{28}$$

where

$$\bar{\mathbf{z}}_{\mathrm{new}} = \bar{\beta}_\mathbf{r}\mathbf{\Sigma}_{\mathbf{z}\mathrm{new}}^{-1}\overline{\mathbf{W}}_\mathbf{r}'\mathbf{r}_{\mathrm{new}}, \tag{29}$$

$$\mathbf{\Sigma}_{\mathbf{z}\mathrm{new}} = \bar{\beta}_\mathbf{r}\left(\overline{\mathbf{W}}_\mathbf{r}'\overline{\mathbf{W}}_\mathbf{r} + \mathbf{\Sigma}_{\mathbf{w_r}}^{-1}\right) + \mathbf{E}. \tag{30}$$

Finally, the predictive distribution is obtained by

$$P(\mathbf{I}_{\text{new}}|\mathbf{r}_{\text{new}}) \simeq \int d\mathbf{z}_{\text{new}} P(\mathbf{I}_{\text{new}}|\mathbf{z}_{\text{new}}) \widetilde{\mathcal{Q}}_{\mathbf{z}}(\mathbf{z}_{\text{new}})$$
$$= \mathcal{N}(\mathbf{I}_{\text{new}}|\bar{\mathbf{I}}_{\text{new}}, \mathbf{\Sigma}_{\mathbf{I}\text{new}}^{-1}), \tag{31}$$

where

$$\bar{\mathbf{I}}_{\text{new}} = \bar{\beta}_{\mathbf{r}} \overline{\mathbf{W}}_{\mathbf{I}} \mathbf{\Sigma}_{\mathbf{z}\text{new}}^{-1} \overline{\mathbf{W}}_{\mathbf{r}}' \mathbf{r}_{\text{new}}, \tag{32}$$
$$\mathbf{\Sigma}_{\mathbf{I}\text{new}} = \overline{\mathbf{W}}_{\mathbf{I}} \mathbf{\Sigma}_{\mathbf{z}\text{new}}^{-1} \overline{\mathbf{W}}_{\mathbf{I}}' + \bar{\beta}_{\mathbf{I}}^{-1} \mathbf{E}. \tag{33}$$

The reconstructed visual image is calculated by taking the expectation of the predictive distribution.

## 2.5 fMRI data

We used the data set from Miyawaki et al. [6], in which fMRI signals were measured while subjects viewed visual images consisting of contrast-defined $10 \times 10$ patches. The data set contained two independent sessions. One is a "random image session", in which spatially random patterns were sequentially presented for 6 s followed by a 6 s rest period. A total of 440 different random patterns were presented for each subject. The other is a "figure image session", in which alphabetical letters and simple geometric shapes were sequentially presented for 12 s followed by a 12 s rest period. Five alphabetical letters and five geometric shapes were presented six or eight times per subject. We used fMRI data from V1 for the analyses. See Miyawaki et al. [6] for details.

# 3 Results

We estimated image bases and weight vectors using the data from the "random image session". Then, reconstruction performance was evaluated with the data from the "figure image session".

## 3.1 Estimated image bases

Figure 2 (a) shows representative image bases estimated by Bayesian CCA (weight values are indicated by a gray scale). The estimation algorithm extracted spatially localized image bases whose shapes were consistent with those used in the previous study [6] ($1 \times 1$, $1 \times 2$, and $2 \times 1$ shown in 1st and 2nd row of Figure 2 (a)). We also found image bases with other shapes (e.g., L-shape, $3 \times 1$ and $1 \times 3$, 3rd row of Figure 2 (a)) that were not assumed in the previous study. We repeated the estimation using data resampled from the random image session, and calculated the distribution of the image bases (defined by a pixel cluster with magnitudes over 3 SD of all pixel values) over eccentricity for different sizes (Figure 2 (a), right). The image bases of the smallest size ($1 \times 1$) were distributed over the visual field, and most of them were within three degrees of eccentricity. The size of the image basis tended to increase with eccentricity. For comparison, we also performed the image basis estimation using CCA, but it did not produce spatially localized image bases (Figure 2 (b)). Estimated weight vectors for fMRI voxels had high values around the retinotopic region corresponding the location of the estimated basis (data not shown).

## 3.2 Visual image reconstruction using estimated image bases

The reconstruction model with the estimated image bases was tested on five alphabet letters and five geometric shapes (Figure 3 (a), 1st row). The images reconstructed by Bayesian CCA captured the essential features of the presented images (Figure 3 (a), 2nd row). In particular, they showed fine reconstruction for figures consisting of thin lines such as small frames and alphabet letters. However, the peripheral reconstruction was poor and often lacked shapes of the presented images. This may be due to the lack of estimated image bases in the peripheral regions (Figure 2 (a), right). The standard CCA produced poorer reconstruction with noise scattered over the entire image (Figure 3 (a), 3rd row), as expected from the non-local image bases estimated by CCA (Figure 2 (b)). Reconstruction using fixed image bases [6] showed moderate accuracy for all image types (Figure 3 (a), 4th row). To evaluate the reconstruction performance quantitatively, we calculated the spatial correlation between the presented and reconstructed images (Figure 3 (b)). The correlation values

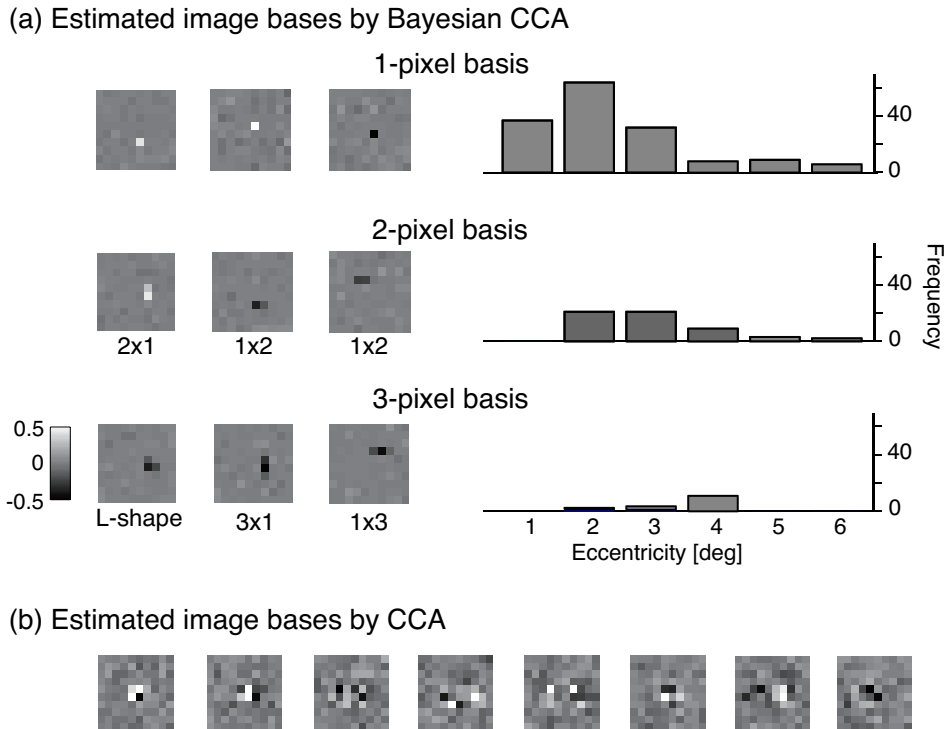

(a) Estimated image bases by Bayesian CCA

Figure 2: **Image basis estimation**: (a) Representative bases estimated by Bayesian CCA (left, sorted by the number of pixels), and their frequency as a function of eccentricity (right). 3-pixel bases (L-shape, 3x1 and 1x3) were not assumed in Miyawaki et al. [6]. Negative (dark) bases were often associated with negative voxel weights, thus equivalent to positive bases with positive voxel weights. (b) Examples of image bases estimated by the standard CCA.

were not significantly different between Bayesian CCA and the fixed basis method when the alphabet letters and the geometric shapes were analyzed together. However, Bayesian CCA outperformed the fixed basis method for the alphabet letters, while the fixed basis method outperformed Bayesian CCA for the geometric shapes ($p < .05$). This is presumably because the alphabet letters consist of more foveal pixels, which overlap the region covered by the image bases estimated by Bayesian CCA. The reconstruction performance of CCA was lowest in all cases.

## 4   Discussion

We have proposed a new method to estimate image bases from fMRI data and presented visual stimuli. Our model consists of the latent variables and two matrices relating the two sets of observations. The previous work used fixed image bases and estimated the weights between the image bases and fMRI voxels. This estimation was conducted by the sparse logistic regression that assumed sparsenes in the weight values, which effectively removed irrelevant voxels [8]. The proposed method introduced sparseness priors not only for fMRI voxels but also for image pixels. These priors lead to automatic extraction of images bases, and the mappings between a small number of fMRI voxels and a small number of image pixels. Using this model, we successfully extracted spatially localized image bases including those not used in the previous work [6]. Using the set of image bases, we were able to accurately reconstruct arbitrary contrast-defined visual images from fMRI activity patterns. The sparseness priors played an important role to estimate spatially localized image bases, and to improve reconstruction performance, as demonstrated by the comparison with the results from standard CCA (Figure 2 and 3).

Our method has several limitations. First, as the latent variables were assumed to have an orthogonal Gaussian distribution, it may be difficult to obtain non-orthogonal image bases, which have been

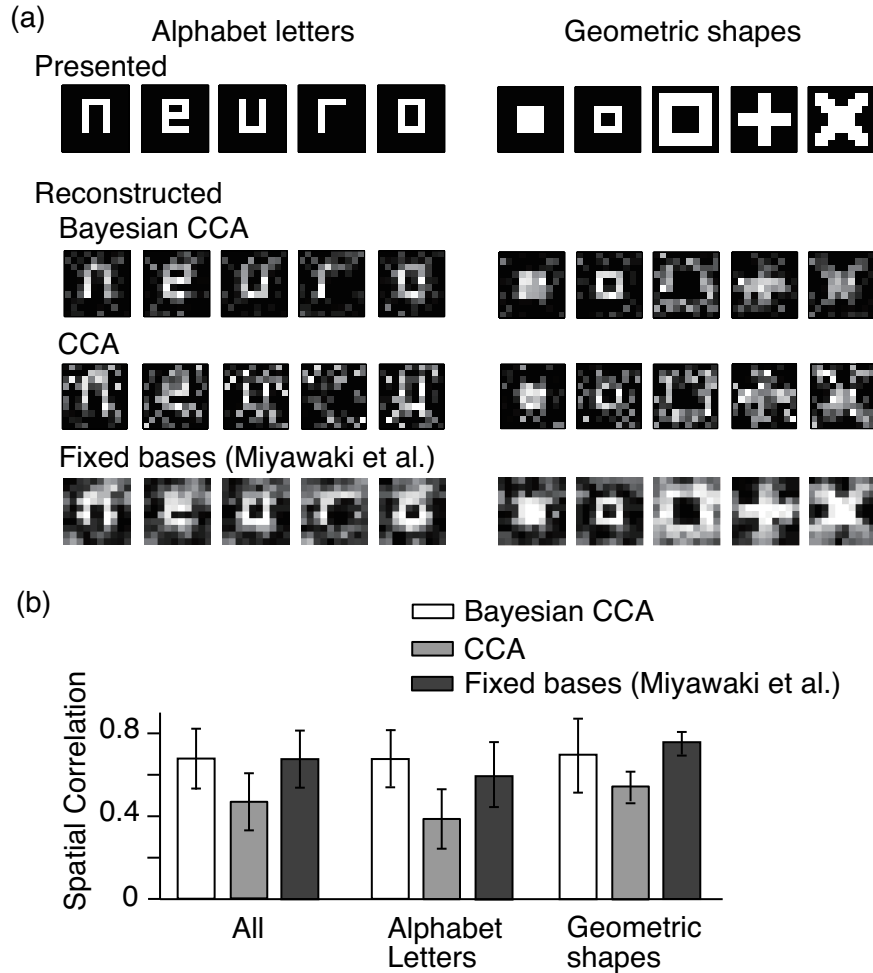

Figure 3: **Visual image reconstruction**: (a) Presented images (1st row, alphabet letters and geometric shapes) and the reconstructed images obtained from Bayesian CCA, the standard CCA, and the fixed basis model (2nd - 4th rows). (b) Spatial correlation between presented and reconstructed images.

shown to provide an effective image representation in the framework of sparse coding [4, 9]. Different types of image bases could be generated by introducing non-orthogonality and/or non-lineality in the model. The shape of estimated image bases may also depend on the visual stimuli used for the training of the reconstruction model. Although we used random images as visual stimuli, other types of images including natural scenes may lead to more effective image bases that allow for accurate reconstruction. Finally, our method failed to estimate peripheral image bases, and as a result, only poor reconstruction was achieved for peripheral pixels. The cortical magnification factor of the visual cortex [5] suggests that a small number of voxels represent a large number of image pixels in the periphery. Elaborate assumptions about the degree of sparseness depending on eccentricity may help to improve basis estimation and image reconstruction in the periphery.

**Acknowledgments**

This study was supported by the Nissan Science Foundation, SCOPE (SOUMU) and SRPBS (MEXT).

# References

[1] Anderson, T.W. (2003). An Introduction to Multivariate Statistical Analysis. 3rd ed. Wiley Interscience.

[2] Attias, H. (1999). Inferring parameters and structure of latent variable models by variational Bayes. Proc. 15th Conference on Uncertainty in Artificial Intelligence, 21-30.

[3] Bach, F.R. and Jordan, M.I. (2005). A probabilistic interpretation of canonical correlation analysis. Dept. Statist., Univ. California, Berkeley, CA, Tech. Repo. 688.

[4] Bell, A.J. and Sejnowski, T.J. (1997) The independent components of natural scenes are edge filter. Vision Res. 27(23), 3327-3338.

[5] Engel, S.A., Glover, G.H. and Wandell, B.A. (1997) Retinotopic organization in human visual cortex and the spatial precision of functional MRI. Cereb. Cortex 7, 181-192.

[6] Miyawaki, Y., Uchida, H., Yamashita, O., Sato, MA., Morito, Y., Tanabe, HC., Sadato, N. and Kamitani, Y. (2008). Visual image reconstruction from human brain activity using a combination of multiscale local image decoders. Neuron 60(5), 915-929.

[7] Neal, R.M. (1996). Bayesian learning for Neural Networks. Springer-Verlag.

[8] Yamashita, O., Sato, MA., Yoshioka, T., Tong, F., Kamitani, Y. (2008) Sparse estimation automatically selects voxels relevant for the decoding of fMRI activity patterns. Neuroimage. 42(4), 1414-29.

[9] Olshausen ,B.A. and Field, D.J. (1996). Emergence of simple-cell receptive field properties by learning a sparse code for natural images. Nature 381, 607-609.

[10] Wang, C. (2007). Variatonal Bayesian Approach to Canonical Correlation Analysis. IEEE Trans Neural Netw. 18(3), 905-910.

